# Absence of Cycles in Symmetric Neural Networks

**Xin Wang**
Computer Science Dept
UCLA
Los Angeles, CA 90024
xwang@cs.ucla.edu

**Arun Jagota, Fernanda Botelho, Max Garzon**
Dept of Mathematical Sciences
University of Memphis
Memphis, TN 38152
jagota, botelhof, garzonm@hermes.msci.memst.edu

## Abstract

For a given recurrent neural network, a discrete-time model may have asymptotic dynamics different from the one of a related continuous-time model. In this paper, we consider a discrete-time model that discretizes the continuous-time leaky integrator model and study its parallel and sequential dynamics for symmetric networks. We provide sufficient (and necessary in many cases) conditions for the discretized model to have the same cycle-free dynamics of the corresponding continuous-time model in symmetric networks.

## 1 INTRODUCTION

For an $n$-neuron recurrent network, a much-studied and widely-used continuous-time (CT) model is the leaky integrator model (Hertz, *et al.*, 1991; Hopfield, 1984), given by a system of nonlinear differential equations:

$$\tau_i \frac{dx_i}{dt} = -x_i + \sigma_i(\sum_{j=1}^{n} w_{ij}x_j + I_i), \quad t \geq 0, \quad i = 1, ..., n, \tag{1}$$

and a related discrete-time (DT) version is the sigmoidal model (Hopfield, 1982; Marcus & Westervelt, 1989), specified by a system of nonlinear difference equations:

$$x_i(t+1) = \sigma_i(\sum_{j=1}^{n} w_{ij}x_j(t) + I_i), \quad t = 0, 1, ..., \quad i = 1, ..., n, \tag{2}$$

where $x_i(t)$, taking values in a compact interval $[a, b]$, represents the state of neuron $i$ at time $t$, $\tau_i$ is the time constant, $W = [w_{ij}]$ is the real-valued weight matrix, $\sigma_i : \Re \to [a, b]$ is the activation function which often takes a sigmoidal form and $I_i$

is the constant external input to neuron $i$. When the network is symmetric (i.e., $W$ is symmetric), the dynamics of both models have been well understood: the CT model (1) is always convergent, namely, every initial state will approach a fixed point asymptotically (Hirsch, 1989; Hertz, *et al.*, 1991; Hopfield, 1984), and the DT model (2) is either convergent or approaches a periodic orbit of period 2 (i.e., a 2-cycle) (Goles, *et al.*, 1985; Marcus & Westervelt, 1989; Koiran, 1994). For results and analyses of fixed points and cycles in networks that are not necessarily symmetric, see (Brown, 1992; Bruck, 1990; Goles, 1986).

For a given symmetric network $(n, W, \sigma_i, I_i)$, the existence of possible 2-cycles in its discrete-time operation is sometimes trouble-some and undesirable, especially in associative memory and neural optimization applications where only fixed points are used to represent memory patterns (Hopfield, 1982) or to encode feasible solutions (Hertz, *et al.*, 1991). Originally in (Hopfield, 1982) a type of sequential dynamics (in which only one randomly chosen neuron updates its state at any time) had to be employed in order to ensure the convergent dynamics of (2). A great deal of work on asymptotic behavior of (2) has focused on constraining the symmetric matrix $W$ so that the model exhibits only convergent dynamics. It was shown in (Goles, *et al.*, 1985) that, for $\sigma_i$ equal to the -1/+1 signum function, if $W$ is positive definite on the set $\{-1, 0, 1\}^n$, then the model (2) is convergent only to fixed points. In (Marcus & Westervelt, 1989), a similar condition on $W$ and neuron gains was derived for networks with differentiable $\sigma_i$'s (see also (Marcus, *et al.*, 1990; Waugh & Westervelt, 1993)). Nevertheless, the fact remains as that not all symmetric networks that are convergent in (1) show the same convergent dynamics in (2).

Such implausibility of the DT model (2) in fully inheriting the dynamics of the CT model (1) leads to study of another DT model in this paper, which generalizes (2) with some new parameters. For symmetric networks, this model has the same types of parallel and sequential dynamics of (2). But, under some conditions on the new parameters (rather than on the weight matrix itself), this model has the same global convergent parallel dynamics and the same local stability around fixed points of (1). Moreover, with these new parameters as bifurcation parameters, the existence of possible 2-cycles can be understood in this model as resulting from the existence of possible period-doubling bifurcation when the parameters are varied. Finally, it is this model, rather than (2), that is used more often in practice as a discrete-time approximation of (1). Based on all of the above, the DT model studied here is a more appropriate discrete-time model of neural networks for purposes of theoretical investigation, numerical simulation and practical application.

## 2   A DISCRETE-TIME MODEL

The DT model that is studied in this paper is

$$x_i(t+1) = (1 - \alpha_i)x_i(t) + \alpha_i \sigma_i (\sum_{j=1}^{n} w_{ij} x_j(t) + I_i), \quad t = 0, 1, ..., \quad i = 1, ..., n, \quad (3)$$

where $\alpha_i$'s are newly introduced parameters, taking values in $(0, 1]$. This model is based on the Euler discretization of the CT model (1) with all $\tau_i = 1$ [1], with $x_i(s)$ and $(x_i(s+1) - x_i(s))/\alpha_i$ approximating $x_i(t)$ and $dx_i(t)/dt$ in (1), respectively, at $t = s * \alpha_i$. It takes the model (2) as its special case of all $\alpha_i = 1$. The new neuron state $x_i(t+1)$ is now a linear combination of the activation function value

$\sigma_i(\sum_{j=1}^n w_{ij}x_j(t) + I_i)$ and the old state $x_i(t)$. Because of $\alpha_i \in (0, 1]$, the model (3) is well defined, in that the iterative maps resulting from the model,

$$F_i(x) = (1 - \alpha_i)x_i + \alpha_i \sigma_i(\sum_{j=1}^n w_{ij}x_j + I_i), \qquad (4)$$

preserve neuron states in the compact interval $[a, b]$.

For the purposes of this paper, neuron activation functions $\sigma_i$ are assumed to satisfy the following constraints:

(i) $\sigma_i$ have continuous first-order derivatives $\sigma_i'(y)$ for all $y \in \Re$;

(ii) $\sigma_i$ are monotone increasing with $\sigma_i'(y) > 0$;

(iii) $\sigma_i'(y) \to 0$ as $y \to \pm\infty$; and

(iv) $\sigma_i'(y)$ take maximal values $\mu_i$, which are usually referred to as *neuron gains*.

Such functions are fairly general, including often-used $[-1, 1]$- and $[0, 1]$-sigmoids, such as $tanh(\mu_i y)$, $2/\pi \tan^{-1}(\pi\mu_i y/2)$, and $1/(1 + e^{-\mu_i y})$. The constraints on $\sigma_i$'s are sufficient for the functions defined by

$$G_i(x_i) = \int_0^{x_i} \sigma_i^{-1}(y)dy. \qquad (5)$$

to have the following properties that will be used subsequently in proofs of several propositions of this paper:

(i) $G_i'(y) = \sigma_i^{-1}(y)$, and particularly $G_i'(\Delta x_i(t)/\alpha_i + x_i(t)) = \sum_j w_{ij}x_i(t) + I_i$;

(ii) $G_i(y) - G_i(z) \le G'(y)(y - z) - 1/(2\mu_i)(y - z)^2 \le G'(y)(y - z)$;

(iii) $G_i''(y) = 1/\sigma_i'(\sigma_i^{-1}(y))$; and

(iv) $G_i'(y_0 + y_1) - G_i'(y_0) \ge (\min_z G_i''(z))y_1 = y_1/\mu_i$,

where $\Delta x_i(t) = x_i(t + 1) - x_i(t)$.

# 3&emsp;PARALLEL AND SEQUENTIAL DYNAMICS

In parallel dynamics, also called synchronous dynamics, all neurons update their states in each time step. In sequential dynamics, a single neuron updates its state in each time step in such a way that each neuron updates its state infinitely-many times, over all time steps $t$. The most widely studied special case of sequential dynamics is called asynchronous dynamics (Hopfield, 1982), in which the neuron whose state is updated is chosen at random. This models asynchronous evolution of a neural network circuit composed of autonomous neurons.

It is easy to see that the discretized DT model (3) shares the same set of fixed points with the CT model (1); that is, a point $x^*$ is a fixed point of (3) (i.e., $x_i^* = F_i(x^*)$ with $F_i$ given in (4)), if and only if it is a fixed point of (1) (i.e., $-x_i^* + \sigma_i(\sum_{ij} w_{ij}x_j^* + I_i) = 0$).

However, as the result of discretization, fixed points may have different asymptotic stability (Wang & Blum, 1992) and periodic points that are not fixed points may occur (Blum & Wang, 1992; Marcus & Westervelt, 1989) in the DT model, especially when all $\alpha_i = 1$. Nevertheless, the discretized DT model retains the same type of the global parallel dynamics and sequential dynamics of (2), as stated in the following two propositions. These results extend the results for all $\alpha_i = 1$ in (Marcus & Westervelt, 1989) to $\alpha_i \in (0, 1]$.

**Proposition 1** *If W is symmetric, any trajectory in parallel dynamics of (3) tends to either a fixed point or a 2-cycle.*

*Proof.* Consider the following function:

$$E(t) = -\sum_{i,j} w_{ij} x_i(t) x_j(t-1) - \sum_i I_i[x_i(t) + x_i(t-1)]$$
$$+ \sum_i (2 - \alpha_i) G_i(x_i(t-1)) + \sum_i \alpha_i G_i(\Delta x_i(t-1)/\alpha_i + x_i(t-1)).$$

When $\alpha_i = 1$, this function is the one used in (Goles, *et al.*, 1985; Marcus & Westervelt, 1989). It can be shown (the details is omitted due to space limitation) that the one-step change of $E(t)$, $\Delta E(t) = E(t+1) - E(t)$, is always less than or equal to 0 and $\Delta E(t) = 0$ implies that the two-step change $\Delta_2 x(t) = x(t+1) - x(t-1)$ is necessarily equal to zero. As $E(t)$ is bounded from below, the network is therefore convergent to either a fixed point or a 2-cycle.  $\square$

**Proposition 2** *If W is symmetric with all $w_{ii} > -(2 - \alpha_i)/(\alpha_i \mu_i)$, the DT model (3) has the sequential dynamics convergent to fixed points for any $\alpha_i \in (0, 1]$.*

*Proof.* Consider the function used in (Hopfield, 1984; Marcus & Westervelt, 1989),

$$L(t) = -\frac{1}{2} \sum_{i,j} w_{ij} x_i(t) x_j(t) - \sum_i I_i x_i(t) + \sum_i G_i(x_i(t)). \qquad (6)$$

If at time $t$ only neuron $i$ is chosen to update its state and all the others remain unchanged, then $L(t)$ is not increasing, and it is strictly decreasing when the one-step change in $x_i(t)$, $\Delta x_i(t)$, is not 0. (The derivation is omitted due to space limitation.) Hence, any sequential trajectory tends to some fixed point.  $\square$

# 4 GLOBAL CONVERGENCE

Call a model of a neural network *cycle-free* if it is globally convergent to fixed points only. The following proposition provides a condition that eliminates the possible "spurious" periodic dynamic behaviors of the discretized DT model (3).

**Proposition 3** *If W is symmetric, a sufficient condition for (3) to be cycle-free in parallel dynamics is*

$$\text{the matrix} \quad W + (2I - A)A^{-1}M^{-1} \quad \text{is positive definite,} \qquad (7)$$

*where $A = diag(\alpha_i)$ and $M = diag(\mu_i)$ are the diagonal matrices formed by the parameters $\alpha_i$ and the neuron gains $\mu_i$.*

*Proof.* Use the energy function $L(t)$ used in the proof of Proposition 2. The one-step difference $\Delta L(t)$ of $L(t)$ along any trajectory $x(t)$ has an upper bound

$$\Delta L(t) \leq -\frac{1}{2} \Delta x(t)^\top (W + (2I - A)A^{-1}M^{-1}) \Delta x(t). \qquad (8)$$

The condition (7) implies that the upper bound is negative and hence the parallel dynamics is globally convergent.  $\square$

In a simple case where all gains $\mu_i = 1$ (e.g., $\sigma_i(z) = \tanh(z)$) and $\alpha_i = \alpha$, this proposition says that the model is cycle-free if the matrix $W + [(2-\alpha)/\alpha]I$ is positive definite.

The sufficient condition (7) generalizes many existing conditions for the cycle-free dynamics in the literature. When $\alpha_i = 1$, it reduces to that matrix $W + M^{-1}$ is positive definite, which is the one presented in (Marcus & Westervelt, 1989) (with all $R_i = 1$ in their model) for the DT model (2) to be cycle-free. Moreover, when $\mu \to \infty$ the sigmoidal functions tend to the signum function. If in this case $\alpha_i \geq \epsilon$ for some fixed positive $\epsilon$, the condition (7) reduces to that the weight matrix $W$ be positive definite, which is the one in (Goles, et al., 1985), except that in the latter case $W$ need be positive definite only on the set $\{-1, 0, 1\}^n$.

When $\alpha_i$ are sufficiently small, the matrix in (7) will be dominated by its positive diagonal entries and become positive definite. In fact,

**Corollary 1** Let $\lambda_{min}$ be the minimum eigenvalue of the symmetric matrix $W$. If either (i) $\lambda_{min} > 0$ (i.e., $W$ is positive definite itself) and $\alpha_i$ are arbitrary in $(0, 1]$, or (ii) $\lambda_{min} \leq 0$ and all $\alpha_i$'s satisfy $(2 - \alpha_i)/(\alpha_i \mu_i) > -\lambda_{min}$, then the model (3) is cycle-free.

*Proof.* Let $W = P^\mathsf{T} \Lambda P$ be an orthogonal decomposition of $W$; that is, $\Lambda$ is a diagonal matrix formed by the eigenvalues of $W$ and $P$ is some orthogonal matrix with its transpose $P^\mathsf{T} = P^{-1}$. The condition (7) is equivalent to that the diagonal matrix

$$\Lambda + (2I - A)A^{-1}M^{-1} \text{ is positive definite.}$$

The later condition can be fulfilled by either condition (i) or (ii). The conclusion then follows from Proposition (3).                                                                             □

This corollary implies that if the weight matrix $W$ is formed according to the Hebb rule as constructed in (Hopfield, 1982), then the model is cycle-free. This is because $W$ is an outer-product $W = VV^\mathsf{T} - mI$ of a collection of some "memory" vectors $V = [v_1, ..., v_m]$, and it is positive definite.

# 5   LOCAL ASYMPTOTIC STABILITY

When all $\alpha_i = \alpha$, the condition (7) in Proposition 3 becomes

$$\text{the matrix } W + \frac{2 - \alpha}{\alpha} M^{-1} \text{ is positive definite.}$$

This is the one given in (Wang & Blum, 1992) that ensures consistency of the DT model (3) with the CT model (1) on local asymptotic dynamics around fixed points for symmetric networks. The consistency means that any fixed point has exactly the same asymptotic stability in both (3) and (1). If these two models are consistent in this regard, a fixed point is an attractor (saddle point or repellor, respectively) of (3) if and only if it is an attractor (saddle point or repellor) of (1). This answers the issue raised in (Marcus & Westervelt, 1989) on why a stable fixed point of (1) is also stable in (2), if a specific version of the condition (7) is met. For symmetric networks, the consistency condition on the local asymptotic dynamics between the CT and DT models turns out to be a consistency condition between them on the global convergent dynamics as well. It is certainly interesting to see if this type of relationship between the local and global consistencies can be extended to general (non-symmetric) networks.

# 6   PERIOD-DOUBLING BIFURCATION

In many cases, the condition (7) in Proposition 3 is also necessary for the network to be cycle-free. This can be addressed from a bifurcation point of view by treating

the parameters $\alpha_i$ as bifurcation parameters. Essentially, the condition (7) gives no room for existence of period-doubling bifurcation, which is the source of generating possible 2-cycles.

**Proposition 4** *Let the activation functions $\sigma_i$ be symmetric, i.e., $\sigma_i : \Re \rightarrow [-a, a]$, and satisfy*

$$\sigma_i(0) = 0, \qquad \sigma_i'(0) = \mu_i.$$

*Let the external bias vector $I = 0$. Then condition (7) is also a necessary condition for the network to be cycle-free.*

*Proof.* Define $C = \{(\alpha_1, \ldots, \alpha_n) \mid W + (2I - A)A^{-1}M^{-1}$ is positive definite$\}$. Let $C_i$ denote the projection of the $i^{th}$ components of the $n$-tuples in $C$. Because $\alpha < \alpha' \in C_i$ implies $\alpha \in C_i$, each $C_i$ is either the entire interval $(0, 1]$ or an open interval $(0, c_{i_0})$ for some $0 < c_{i_0} < 1$. Notice that 0 is a fixed point of the network.

The Jacobian of the iterative maps in (4) at the fixed point 0 is

$$(I - A) + AMW. \tag{9}$$

Notice that the condition (7) is equivalent to that the eigenvalues of $(2I - A) + AMW$ are all positive, which is further equivalent to that the Jacobian (9) has all eigenvalues $\lambda \geq -1$.

If $C = (0, 1]^n$, the model has no cycles, according to Proposition 3, for any $(\alpha_1, \ldots, \alpha_n) \in (0, 1]^n$. However, if $C_i = (0, c_{i_0})$ for some $i$ and $c_{i_0} < 1$, some eigenvalue of the Jacobian (9) becomes less than $-1$ when $\alpha_i$ exceeds the "threshold" $c_{i_0}$. During this course of changing $\alpha_i$, the network undergoes generically a period-doubling bifurcation (Ruelle, 1989), resulting in emergence of some 2-cycles. Thus, in this case the condition (7) in Proposition 3 is also necessary to prevent this type of period-doubling bifurcation from happening around fixed points and hence to eliminate possibility of generating 2-cycles. $\square$

Examples of $\sigma_i$'s satisfying hypotheses of Proposition 4 are $\tanh_{\mu_i} : \Re \rightarrow [-1, 1]$ with $\tanh_{\mu_i}(z) = \tanh(\mu_i z)$.

# 7  EFFECT OF NEURON GAINS IN NEURAL COMPUTATIONS

Considerable research has been conducted on using (1) in neural computations such as solving optimization problems approximately; see (Hertz, *et al.*, 1991, Chapter 4) for an overview. Often, the neuron gains $\mu_i$ are also modified while the network is evolving. A popular algorithm of this kind uses mean field annealing (MFA) (Peterson & Anderson, 1988) to solve optimization problems, in which small neuron gains are used initially, and increased gradually. Similar situations also happen in some learning algorithms.

In practice, a discretized model such as (3) is used instead. Proposition 3 gives some criterion on how to choose the "discretization step-sizes" $\alpha_i$ as functions of $\mu_i$. If efficiency, for example, were the paramount consideration, one might want to choose $\alpha_i$ as large as possible while ensuring that the sufficient condition of Proposition 3 is met.

The effect of changing $\mu$ on the largest sufficing $\alpha$ can be examined as follows. For simplicity, consider the case where all neuron gains $\mu_i$ equal $\mu$ and all $\alpha_i$ equal to $\alpha$. Let $c_i$, $i = 1, 2$, be the respective supremums of $\alpha$'s such that $W + (2 - \alpha)/(\alpha d_i)I$ are positive definite when the neuron gains $\mu$ are equal to two different values $d_1$ and

$d_2$. Then $c_1$ and $c_2$ satisfy $(2 - c_1)/(c_1 d_1) = (2 - c_2)(c_2 d_2)$. Letting $\beta = d_2/d_1$, the above gives $c_2 = 2c_1/(c_1 + \beta(2 - c_1))$. Clearly, $c_2$ is proportional to the reciprocal of the ratio $\beta$. Thus, when $\mu$ is small, $\alpha$ can be taken larger than when $\mu$ is large. This may be used to evolve the network efficiently in the beginning and slow it down later, while ensuring that 2-cycles are never retrieved.

## Footnotes

[1] When (1) is globally convergent to fixed points, neglecting all $\tau_i$ does not change its dynamics.

# References

E.K. Blum & X. Wang. (1992) Stability of fixed points and periodic orbits and bifurcations in analog neural networks. *Neural Networks*, 5:577–587.

D.P. Brown. (1992) Matrix tests for period 1 and 2 limit cycles in discrete threshold networks. *IEEE Trans. on Systems, Man, & Cybernetics*, 22:552–554.

J. Bruck. (1990) On the convergence properties of the Hopfield model. *Proc. IEEE*, 78:1579–1585.

E. Goles, F. Fogelman-Soulie & D. Pellegrin. (1985) Decreasing energy functions as a tool for studying threshold networks. *Discrete Applied Mathematics*, 12:261–277.

E. Goles. (1982) Fixed point behaviour of threshold functions on a finite set. *SIAM J. of Algorithmic Discrete Methods*, 3:529–531.

E. Goles. (1986) Antisymmetrical neural networks. *Discrete Applied Mathematics*, 13:97–100.

J. Hertz, A. Krogh & R.G. Palmer. (1991) *Introduction to the Theory of Neural Computation*. Addison-Wesley.

M.W. Hirsch. (1989) Convergent activation dynamics in continuous time networks. *Neural Networks*, 2:331–349.

J.J. Hopfield. (1982) Neural networks and physical systems with collective computational abilities. *Proc. of the National Academy of Sciences, USA*, 79:2554–2558.

J.J. Hopfield. (1984) Neurons with graded response have collective computational properties like those of two-state neurons. *Proc. of the National Academy of Sciences, USA*, 81:3088–3092.

P. Koiran. (1994) Dynamics of discrete time, continuous state Hopfield networks. *Neural Computation*, 6:459–468.

C.M. Marcus & R.M. Westervelt. (1989) Dynamics of iterated-map neural networks. *Physical Review A*, 40:501–504.

C. Peterson & J.R. Anderson. (1988) Neural networks and NP-complete optimization problems; a performance study on the graph bisection problem. *Complex Systems*, 2:59–89.

D. Ruelle. (1989) *Elements of Differentiable dynamics and Bifurcation Theory*. Academic Press, Inc.

X. Wang & E.K. Blum. (1992) Discrete-time versus continuous-time neural networks. *J. of Computer and System Sciences*, 49:1–17.

F.R. Waugh & R.M. Westervelt. (1993) Analog neural networks with local competition. I. Dynamics and stability. *Physical Review E*, 47:4524–4536.
